# Human memory search as a random walk in a semantic network

**Joshua T. Abbott**
Department of Psychology
University of California, Berkeley
Berkeley, CA 94720
joshua.abbott@berkeley.edu

**Joseph L. Austerweil**
Department of Psychology
University of California, Berkeley
Berkeley, CA 94720
joseph.austerweil@gmail.com

**Thomas L. Griffiths**
Department of Psychology
University of California, Berkeley
Berkeley, CA 94720
tom_griffiths@berkeley.edu

## Abstract

The human mind has a remarkable ability to store a vast amount of information in memory, and an even more remarkable ability to retrieve these experiences when needed. Understanding the representations and algorithms that underlie human memory search could potentially be useful in other information retrieval settings, including internet search. Psychological studies have revealed clear regularities in how people search their memory, with clusters of semantically related items tending to be retrieved together. These findings have recently been taken as evidence that human memory search is similar to animals foraging for food in patchy environments, with people making a rational decision to switch away from a cluster of related information as it becomes depleted. We demonstrate that the results that were taken as evidence for this account also emerge from a random walk on a semantic network, much like the random web surfer model used in internet search engines. This offers a simpler and more unified account of how people search their memory, postulating a single process rather than one process for exploring a cluster and one process for switching between clusters.

## 1 Introduction

Human memory has a vast capacity, storing all the semantic knowledge, facts, and experiences that people accrue over a lifetime. Given this huge repository of data, retrieving any one piece of information from memory is a challenging computational problem. In fact, it is the same problem faced by libraries [1] and internet search engines [6] that need to efficiently organize information to facilitate retrieval of those items most likely to be relevant to a query. It thus becomes interesting to try to understand exactly what kind of algorithms and representations are used when people search their memory.

One of the main tasks that has been used to explore memory search is the semantic fluency task, in which people retrieve as many items belonging to a particular category (e.g., animals) as they can in a limited time period. Early studies using semantic fluency tasks suggested a two-part memory retrieval process: *clustering*, in which the production of words form semantic subcategories, and *switching*, in which a transition is made from one subcategory to another [13, 21]. This decomposition of behavior has been useful for diagnosing individual participants with particular clinical

conditions such as Alzheimer's and Parkinson's disease, which result in different patterns of deficits in these processes [9, 22].

Recently, it has been suggested that the clustering patterns observed in semantic fluency tasks could reflect an optimal foraging strategy, with people searching for items distributed in memory in a way that is similar to animals searching for food in environments with patchy food resources [7]. The idea behind this approach is that each cluster corresponds to a "patch" and people strategically choose to leave patches when the rate at which they retrieve relevant concepts drops below their average rate of retrieval. Quantitative analyses of human data provide support for this account, finding shorter delays in retrieving relevant items after a change in clusters and a relationship between when people leave a cluster and their average retrieval time.

In this paper, we argue that there may be a simpler explanation for the patterns seen in semantic fluency tasks, requiring only a single cognitive process rather than separate processes for exploring a cluster and deciding to switch between clusters. We show that the results used to argue for the optimal foraging account can be reproduced by a random walk on a semantic network derived from human semantic associations. Intriguingly, this is exactly the kind of process assumed by the PageRank algorithm [12], providing a suggestive link between human memory and internet search and a new piece of evidence supporting the claim [6] that this algorithm might be relevant to understanding human semantic memory.

The plan of the paper is as follows. Section 2 provides relevant background information on studies of human memory search with semantic fluency tasks and outlines the retrieval phenomena predicted by an optimal foraging account. Section 3 presents the parallels between searching the internet and search in human memory, and provides a structural analysis of semantic memory. Section 4 evaluates our proposal that a random walk in a semantic network is consistent with the observed behavior in semantic fluency tasks. Finally, Section 5 discusses the implications of our work.

## 2  Semantic fluency and optimal foraging

Semantic fluency tasks (also known as free recall from natural categories) are a classic methodological paradigm for examining how people recall relevant pieces of information from memory given a retrieval cue [2, 14, 19]. Asking people to retrieve as many examples of a category as possible in a limited time is a simple task to carry out in clinical settings, and semantic fluency has been used to study memory deficits in patients with Alzheimer's, Parkinson's, and Huntington's disease [9, 20, 21, 22]. Both early and recent studies [2, 14, 21] have consistently found that clusters appear in the sequences of words that people produce, with bursts of semantically related words produced together and noticeable pauses between these bursts. For example Troyer et al. [21] had people retrieve examples of animals, and divided those animals into 22 nonexclusive clusters ("pets", "African animals", etc.). These clusters could be used to analyze patterns in people's responses: if an item shares a cluster with the item immediately before it, it is considered part of the same cluster, otherwise, the current item defines a transition between clusters. For example, given the sequence "dog-cat-giraffe", "dog" and "cat" are considered elements of the same cluster, while "giraffe" is considered a point of transition to a new patch. Observing fast transitions between items within a cluster but slow transitions between clusters led to the proposal that memory search might be decomposed into separate "clustering" and "switching" processes [21].

The clusters that seem to appear in semantic memory suggest an analogy to the distribution of animal food sources in a patchy environment. When animals search for food, they must consider the costs and benefits of staying within a patch as opposed to searching for a new patch. Optimal foraging theory [16] explores the ideal strategies for solving this problem. In particular, the *marginal value theorem* shows that a forager's overall rate of return is optimized if it leaves a patch when the instantaneous rate (the marginal value) of finding food within the patch falls below the long-term average rate of finding food over the entire environment [3]. In a recent proposal, Hills et al. [7] posited that search in human semantic memory is similarly guided by an optimal foraging policy. The corresponding prediction is that people should leave a "patch" in memory (ie. a semantically related cluster) when the the marginal value of resource gain (finding more relevant items) falls below the expected rate of searching elsewhere in memory.

To investigate these predictions, Hills et al. [7] had people perform a semantic fluency task ("Name as many animals as you can in 3 minutes") and analyzed the search paths taken through memory

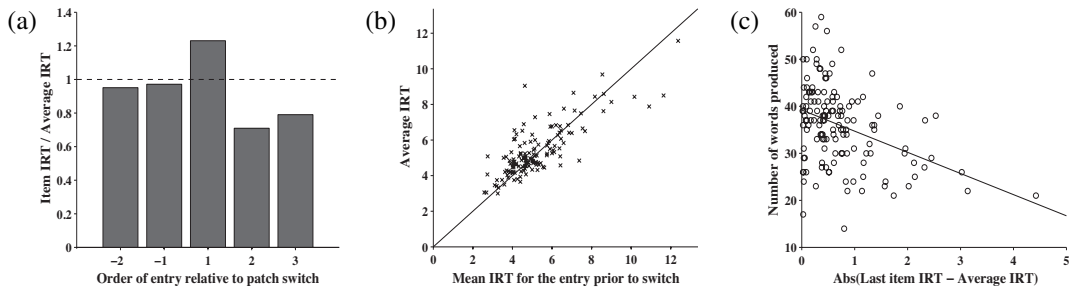

Figure 1: Human results from the Hills et al. [7] animal naming task. (a) The mean ratio between the inter-item response time (IRT) for an item and the participant's long-term average IRT over the entire task, relative to the order of entry for the item (where "1" refers to the relative IRT between the first word in a patch and the last word in the preceding patch). The dotted line indicates where item IRTs would be the same as the participant's average IRT for the entire task. (b) The long-term average IRT versus the mean IRT prior to a switch for each participant. (c) The relationship between a participant's deviation from the marginal value theorem policy for patch departures (horizontal-axis) and the total number of words a participant produced.

in terms of the sequences of animal names produced, assessed with the predetermined animal sub-categories of Troyer et al. [21]. As a first measure of correspondence with optimal foraging theory, the ratio between inter-item response times (IRTs) of items and the long-term average IRTs for each participant were examined at different retrieval positions relative to a patch switch. Figure 1 (a) displays the results of this analysis. The first word in a patch (indicated by an order of entry of "1") takes longer to produce than the overall long-term average IRT (indicated by the dotted line), and the second word in a patch (indicated by "2") takes much less time to produce. These results are in line with the marginal value theorem where IRTs up until a patch switch should increase monotonically towards the long-term average IRT and go above this average only for patch switch IRTs since it takes extra time to find a new patch. Hills et al. offered a two-part process model to account for this phenomenon: When the IRT following a word exceeds the long-term average IRT, search switches from local to global cues (e.g. switching between using semantic similarity or overall frequency as search cues).

To formally examine how close the IRTs for words immediately preceding a patch switch were to the long-term average IRT, the per-participant average IRT for these pre-switch words was plotted against the per-participant long-term average IRT (see Figure 1 (b)). The difference between these IRTs is very small, with a majority of participant's pre-switch IRTs taking less time than their long-term average IRT as predicted by the marginal value theorem. As a further analysis of these pre-switch IRTs, the absolute difference between the pre-switch IRT and long-term average IRT was plotted against the number of words a participant produced along with a regression line through this data (see Figure 1 (c)). Participants with a larger absolute difference (indicating they either left patches too soon or too late) produced fewer words, as predicted by the marginal value theorem.

## 3 The structure of semantic memory

The explanation proposed by Hills et al. [7] for the patterns observed in people's behavior in semantic fluency tasks is relatively complex, assuming two separate processes and a strategic decision to switch between them. In the remainder of the paper we consider a simpler alternative explanation, based on the structure of semantic memory. Specifically, we consider the consequences of a single search process operating over a richer representation – a semantic network.

A semantic network represents the relationships between words (or concepts) as a directed graph, where each word is represented as a node and nodes are connected together with edges that represent pairwise association [4]. Semantic networks derived from human behavior can be used to explore questions about the structure of human memory [5, 14, 17, 18]. We will focus on a network derived from a word association task, in which people were asked to list the words that come to mind for a particular cue. For example, when given the cue "doctor", a person might produce the associates

"nurse", "hospital", and "sick" [11]. This task was repeated with a large number of participants, with each response that was produced more than once being used as a cue in turn. The result is a semantic network with 5018 nodes, from "a" to "zucchini".

If the clusters that appear in people's responses in the semantic fluency task are reflected in the structure of this semantic network, a simple process that moves around the semantic network without explicitly knowing that it contains clusters might be sufficient to capture the phenomena reported by Hills et al. [7]. We explored whether the distance between the nodes corresponding to different animals in the semantic network could be predicted by their cluster membership. The 141 participants in the study conducted by Hills et al. produced 373 unique animals, of which 178 were included in the semantic network. However, 13 of these were "sources", not having been produced as associates for any other words, and we eliminated these from our analysis (as well as the other analyses we report later in the paper). The result was a set of 165 nodes that each had incoming and outgoing edges. We analyzed whether the relationship between these animals in the semantic network showed evidence of the clustering seen in semantic fluency tasks, based on the clusters identified by Troyer et al. [21].

Our analysis was performed using an additive clustering model [15]. Letting $\mathbf{S}$ be the $165 \times 165$ matrix of similarities obtained by taking $s_{ij} = \exp\{-d_{ij}\}$, where $d_{ij}$ is the length of the shortest path between animal nodes $i$ and $j$ in the semantic network, the similarity matrix according to additive clustering is

$$\mathbf{S} = \mathbf{FWF}'$$ (1)

where $\mathbf{F}$ is a feature matrix ($f_{ac} = 1$ if animal $a$ has feature $c$) and $\mathbf{W}$ is a diagonal matrix of (non-negative) cluster weights. The features in the matrix $\mathbf{F}$ were defined to be the twenty-two hand-coded subcategorization of animals from Troyer et al. [21], and $\mathbf{W}$ was found by maximizing the posterior distribution over weights obtained by assuming Gaussian error in reconstructing $\mathbf{S}$ and a Gaussian prior on $\mathbf{W}$ (as in [10]).

The empirical similarity matrix $\mathbf{S}$ and its reconstruction using the clusters are shown in Figure 2 (a) and (b) respectively. The two similarity matrices contain similar block structure, which supports the hypothesis that the clusters of animals are implicitly captured by the semantic network. If the distance between animals in different clusters is greater than the distance between animals in the same cluster, as these results suggest, then a simple search process that is sensitive to this distance may be able to account for the results reported by Hills et al. [7].

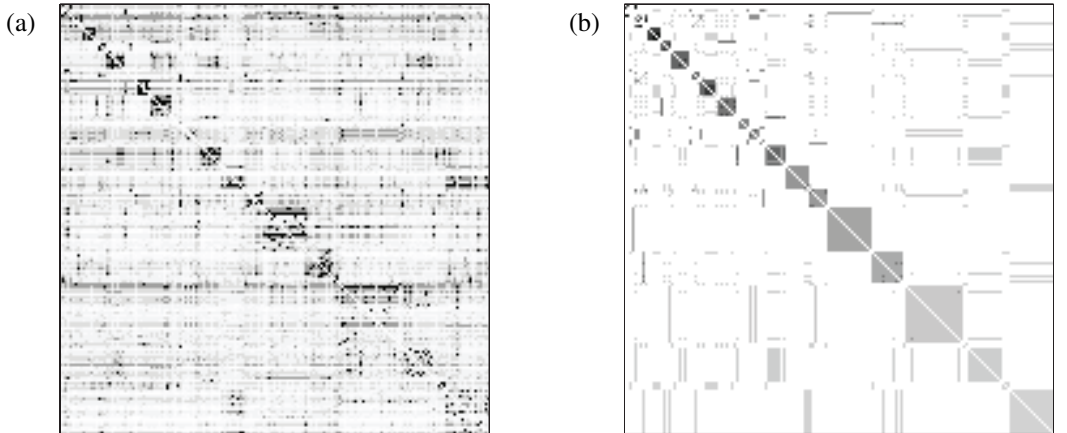

Figure 2: Visualizing the similarity between pairs of animals in our semantic network (darker colors represent stronger similarities). (a) Similarity matrix derived by exponentiating the negative shortest path distance between each pair of animals. (b) Similarity matrix obtained using the additive clustering model where the features are the Troyer et al. [21] clusters and weights are inferred using Nelder-Mead simplex search [8]. The rows and columns of the two matrices were reordered to display animals in the clusters with largest weight first.

# 4 Random walks and semantic fluency

One of the simplest processes that can operate over a semantic network is a random walk, stochastically jumping from one node to another by following edges. Intuitively, this might provide a reasonable model for searching through semantic memory, being a meandering rather than a directed search. Random walks on semantic networks have previously been proposed as a possible account of behavior on fluency tasks: Griffiths et al. [6] argued that the responses that people produce when asked to generate a word that begins with a particular letter were consistent with the stationary distribution of a random walk on the same semantic network used in the analysis presented in the previous section.

In addition to being simple, random walks on semantic networks have an interesting connection to methods used for information retrieval. The PageRank algorithm [12], a component of the original Google search engine, considers web pages as nodes and links as directed edges from one node to another. The PageRank algorithm is the result of a simple observation about web pages (and more broadly, any directed graph): important web pages are linked to by other important web pages. The link structure of $n$ web pages on the Internet can be characterized by an $n \times n$ matrix $\mathbf{L}$, where $\mathbf{L}_{ij}$ is 1 if there is a link from web page $j$ to web page $i$, and 0 otherwise. If an internet user clicks uniformly at random over the outgoing links, then the probability that the user will click on page $i$ given she is currently on page $j$ is

$$\mathbf{M}_{ij} = \frac{\mathbf{L}_{ij}}{\sum_{k=1}^{n} \mathbf{L}_{kj}} \tag{2}$$

where the denominator is the *out-degree* or number of web pages that page $j$ links to. Thus, $\mathbf{M}$ is the transition matrix of a Markov chain and under mild conditions, the probability that a "random surfer" will be on any page regardless of where she starts is given by the vector $\mathbf{p}$ that solves $\mathbf{p} = \mathbf{Mp}$. This is the eigenvector of $\mathbf{M}$ corresponding to its largest eigenvalue (which is 1 as $\mathbf{M}$ is a stochastic matrix).

Viewed in this light, the finding reported by Griffiths et al. [6] is that the prominence of words in human memory can be predicted by running the PageRank algorithm on a semantic network. However, as Griffiths et al. pointed out, multiple mechanisms exist that could produce this result, with only one possibility being that memory search is a random walk on a semantic network. Exploring whether this kind of random walk can reproduce the phenomena identified by Hills et al. [7] in a completely different memory task would provide further support for this possibility.

In the remainder of this section, we explore some variations on a simple random walk that result in four different models. We then evaluate our models of memory search by applying the analyses used by Hills et al. [7] to their behavior.

## 4.1 Random walk models

In the experiment reported by Hills et al. [7] participants were asked to produce as many unique animals as possible in three minutes. A simple generative model for this sequential process is a Markov chain that starts at state $X_0 = $ "animal", and then at step $n$ randomly generates the next state $X_{n+1}$ according to a probability distribution that only depends on the current state $X_n$ (and possibly the cue $C = $ "animal"). We define a space of four possible models by varying two dimensions for how we define the transition probabilities.

The first dimension is the transition model, which can either be *uniform*, where the next state is chosen uniformly at random from the outgoing links of the current node (ie. using the transition matrix $\mathbf{M}$ defined above), or *weighted*, where the probability of the next state is weighted according to the frequency of transitions in the word-association data [11]. This captures the fact that stronger associations (e.g., "cat" and "mouse") are produced more frequently than weaker associations (e.g., "cat" and "house"), even though "cat" was produced given either word.

The second dimension is the effect of the cue at each step, which was either *non-jumping* (it has no effect except for initializing the chain at "animal") or *jumping*[1], where the cue causes us to

jump back to "animal" and transition from there, $P(X_{n+1}|X_n = \text{"animal"})$, with probability $\rho$ (but otherwise transition normally with probability $1-\rho$). A jumping process is actually also a part of the PageRank algorithm, which incorporates modifications to the graph that are equivalent to randomly restarting the random surfer in order to deal with violations of ergodicity [12].

Formally, the space of models is defined by

$$P(X_{n+1}|C = \text{"animal"}, X_n = x_n) = \rho P(X_{n+1}|X_n = \text{"animal"}) + (1-\rho)P(X_{n+1}|X_n = x_n) \tag{3}$$

where $P(X_{n+1}|X_n)$ is either *uniform* or *weighted*, and $\rho = 0$ is *non-jumping* or $0 < \rho \leq 1$ is *jumping*.

## 4.2 Computing inter-item retrieval times

Random walk simulations for the models defined above will produce a list of the nodes visited at each iteration. A method of mapping this output to reaction times is necessary in order to make an appropriate comparison with human results. In our analyses we consider only the first time an animal node is visited, which we denote as $\tau(k)$ for the $k^{\text{th}}$ unique animal seen on the random walk (out of the $K$ unique animals seen on the random walk). For example, a simulation may produce the following output:

$$X_1 = \text{"animal"}, X_2 = \text{"dog"}, X_3 = \text{"house"}, X_4 = \text{"dog"}, X_5 = \text{"cat"}.$$

Here, $K = 2$ with $k = 1$ and $k = 2$ referring to "dog" and "cat" respectively. Our $\tau(k)$ function would return $\tau(1) = 2$ and $\tau(2) = 5$ for this example since we only care about the first time "dog" is visited (at timestep $n = 2$) and "house" (at timestep $n = 3$) is not an animal.

An additional assumption that we made is that the amount of time the Markov chain spends to "emit" an animal is the length of the word. As participants in Hills et al. [7] typed their responses, this accounts for it taking longer for participants to type longer than shorter words. Thus, according to the random walk models, the inter-item retrieval time (IRT) between animal $k$ and $k-1$ is

$$IRT(k) = \tau(k) - \tau(k-1) + L(X_{\tau(k)}) \tag{4}$$

where $\tau(k)$ is the first hitting time of animal $X_{\tau(k)}$ and $L(X)$ is the length of word $X$. In our example above, the IRT between "cat" ($k = 2$) and "dog" ($k = 1$) is:

$$IRT(\text{"cat"}) = \tau(\text{"cat"}) - \tau(\text{"dog"}) + L(\text{"cat"}) = 5 - 2 + 3 = 6.$$

With this mapping defined, we can now perform the same set of analyses in Hills et al. [7] on IRTs between animal words for our random walker simulations.

## 4.3 Evaluating the models

We ran 1000 simulations of each of the four models for a duration of 1750 iterations. The number of iterations was selected to produce a similar mean number of animals to those produced by participants in Hills et al. [7]. Human participants produced an average of 36.8 animals, while the uniform non-jumping, uniform jumping, weighted non-jumping, and weighted jumping models produced an average of 30.6, 39.3, 21.0, and 29.1 animals respectively.[2] The jumping models had a probability of $\rho = 0.05$ of making a jump back to "animals", selected primarily to illustrate the impact of adding this additional component to the search process.

All four models were subjected to the same analyses as Hills et al. [7] applied to the human data (Figure 1). The model results are presented in Figure 3. The left column shows the mean ratio between the inter-item retrieval time (IRT) for an item and the mean IRT over all 1750 iterations in the simulations, relative to the order of entry for the item. Here we see that the first word starting a patch (the bar labeled "1") has the highest overall retrieval time. This was interpreted by Hills et al. as indicating the time it takes to switch clusters and generate a word from a new cluster. The

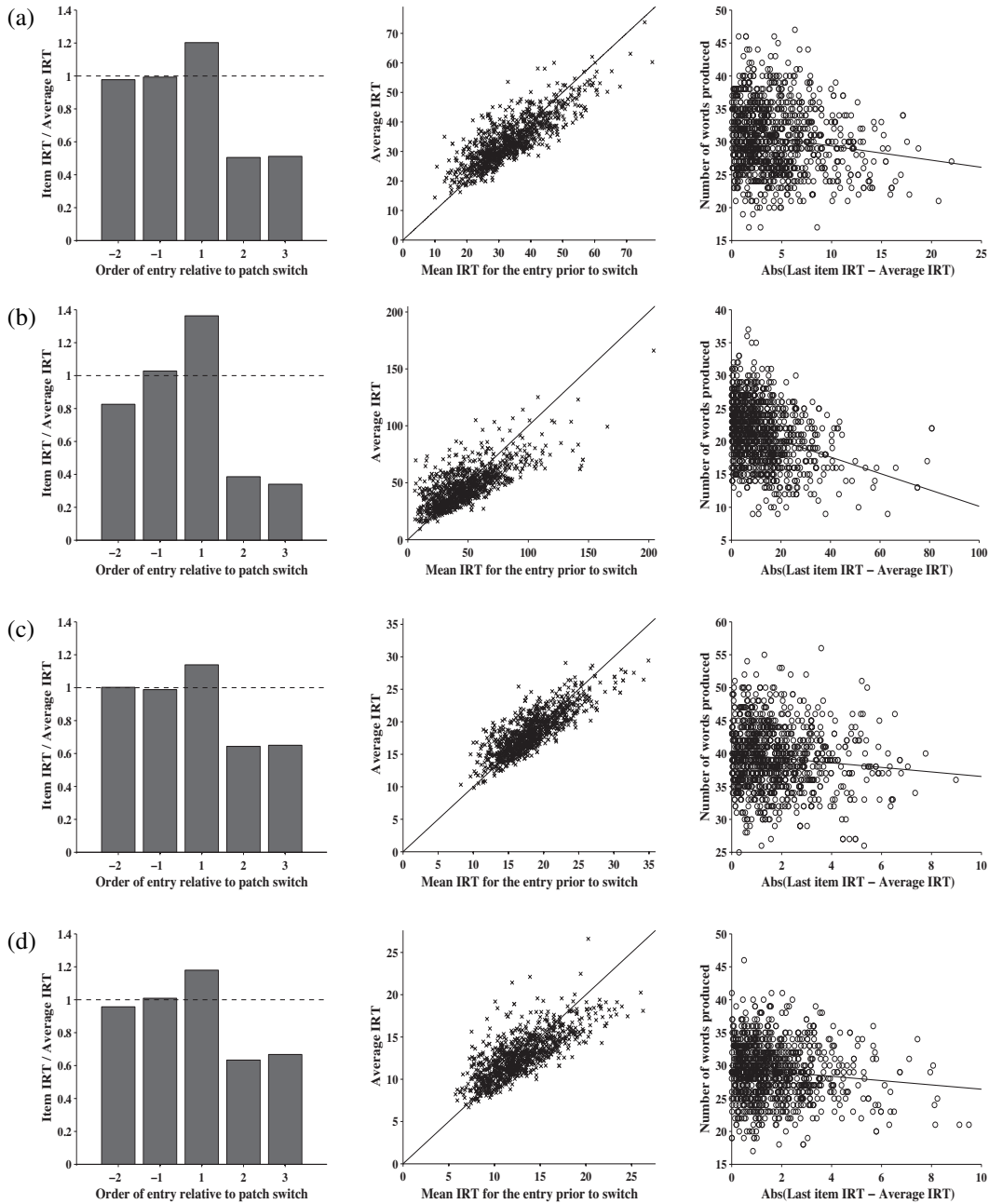

Figure 3: The model results after 1000 simulations of the four random walk models: (a) the uniform transition model with no jumps, (b) the non-uniform transition model with no jumps, (c) the uniform transition model with a jump probability of 0.05, and (d) the non-uniform transition model with a jump probability of 0.05. The left-most column displays the mean ratio between the inter-item retrieval time (IRT) for an item and the long-term average IRT over the entire task, for each model simulation, relative to the order of entry for the item (where "1" refers to the relative IRT between the first word in a patch and the last word in the preceding patch). The dotted line indicates where item IRTs would be the same as a simulation's average IRT for the entire task. The middle column displays the long-term average IRT versus a simulation's mean IRT prior to a switch. The right-most column displays the relationship between a simulation's deviation from the marginal value theorem policy for patch departures (horizontal axis) and the total number of words the simulation produced.

emergence of the same phenomenon is seen across all four of our models which suggests that the structure of semantic memory, together with a simple undirected search process, is sufficient to capture this effect. The introduction of jumps primarily reduces the difference between the IRTs before and after a cluster switch. Additionally, we reproduced the same statistical tests as Hills et al. [7] on the models, demonstrating that, like people, all four models take a significantly longer amount of time for the word immediately following a patch (all $t(999) > 44, p < 0.0001$) and take a significantly shorter amount of time for the second item after a patch (all $t(999) < -49, p < 0.0001$).

The second and third columns of Figure 3 show how the simulated results produced by the four models relate to the predictions of the marginal value theorem. Intriguingly, all four models produce the basic phenomena taken as evidence for the use of the marginal value theorem in memory search. There is a strong correlation between the IRT at the point of a cluster switch and the mean IRT ($R^2 = 0.67, 0.67, 0.52$ and $0.57$ for the four models in the order of Figure 3, all $F(1, 998) > 1000, p < 0.0001$), and a negative relationship between acting in the way stipulated by the marginal value theorem and the number of responses produced ($R^2 = 0.02, 0.01, 0.10$, and $R^2 = 0.01$ for the four models in the same order as before, and all $F(1, 998) > 10, p < 0.001$). These results show that behavior consistent with following the marginal value theorem can be produced by surprisingly simple search algorithms, at least when measured along these metrics.

## 5  Discussion

Understanding how people organize and search through information stored in memory has the potential to inform how we construct automated information retrieval systems. In this paper, we considered two different accounts for the appearance of semantically-related clusters when people retrieve a sequence of items from memory. These accounts differ in the number of processes they postulate and in the rationality they attribute to those processes. The idea that human memory search might follow the principles of optimal foraging [7] builds on previous work suggesting that there are two separate processes involved in semantic fluency tasks – generating from a cluster and switching between clusters [21] – and views the shift between processes as being governed by the rational principles embodied in the marginal value theorem. In contrast, the proposal that memory search might just be a random walk on a semantic network [6] postulates a single, undirected process. Our results show that four random walk models qualitatively reproduce a set of results predicted by optimal foraging theory, providing an alternative explanation for clustering in semantic fluency tasks.

Finding that a random walk on a semantic network can account for some of the relatively complex phenomena that appear in the semantic fluency task provides further support for the idea that memory search might simply be a random walk. This result helps to clarify the possible mechanisms that could account for PageRank predicting the prominence of words in semantic memory [6], since PageRank is simply the stationary distribution of the Markov chain defined by this random walk. This simple mechanism seems particularly attractive given its existing connections to ideas that appear in the information retrieval literature.

Demonstrating that the random walk models can produce behavior consistent with optimal foraging in semantic fluency tasks generates some interesting directions for future research. Having two competing accounts of the same phenomena suggests that the next step in exploring semantic fluency is designing an experiment that distinguishes between these accounts. Considering whether the optimal foraging account can also predict the prominence of words in semantic memory, where the random walk model is already known to succeed, is one possibility, as is exploring the predictions of the two accounts across a wider range of memory search tasks. However, one of the most intriguing directions for future research is considering how these different proposals fare in accounting for changes in semantic fluency in clinical populations. Given that conditions such as Alzheimer's and Parkinson's disease differentially affect clustering and switching [9, 22], considering the different failure conditions of these models might help to answer practical as well as theoretical questions about human memory.

**Acknowledgments.** This work was supported by grants IIS-0845410 from the National Science Foundation and FA-9550-10-1-0232 from the Air Force Office of Scientific Research.

## Footnotes

[1]We note this is a qualitatively different operation than the Hills et al. [7] proposal of "jumping" between different search cues. Instead, this dimension explores the effect of priming the search process by returning to the initial state.

[2]A slightly lower overall total number of animals is to be expected, given the limited number of animals among the words included in our semantic network.

# References

[1] J. R. Anderson. *The adaptive character of thought*. Erlbaum, Hillsdale, NJ, 1990.

[2] W. A. Bousfield and C. H. W. Sedgewick. An analysis of sequences of restricted associative responses. *Journal of General Psychology*, 30:149–165, 1944.

[3] E.L. Charnov et al. Optimal foraging, the marginal value theorem. *Theoretical Population Biology*, 9(2):129–136, 1976.

[4] A. M. Collins and E. F. Loftus. A spreading-activation theory of semantic processing. *Psychological Review*, 82(6):407, 1975.

[5] T. L. Griffiths, M. Steyvers, and J. B. Tenenbaum. Topics in semantic representation. *Psychological Review*, 114:211–244, 2007.

[6] T.L. Griffiths, M. Steyvers, and A. Firl. Google and the mind. *Psychological Science*, 18(12):1069–1076, 2007.

[7] T.T. Hills, M.N. Jones, and P.M. Todd. Optimal foraging in semantic memory. *Psychological Review*, 119(2):431–440, 2012.

[8] J. C. Lagarias, J. A. Reeds, M. H. Wright, and P. E. Wright. Convergence properties of the Nelder-Mead simplex method in low dimensions. *SIAM journal of optimization*, 9:112–147, 1998.

[9] M.D. Lezak. *Neuropsychological assessment*. Oxford University Press, USA, 1995.

[10] D. J. Navarro and T. L. Griffiths. Latent features in similarity judgments: A nonparametric Bayesian approach. *Neural Computation*, 20:2597–2628, 2008.

[11] D.L. Nelson, C.L. McEvoy, and T.A. Schreiber. The University of South Florida free association, rhyme, and word fragment norms. *Behavior Research Methods*, 36(3):402–407, 2004.

[12] L. Page, S. Brin, R. Motwani, and T. Winograd. The PageRank citation ranking: Bringing order to the web. Technical Report 1999-66, Stanford InfoLab, November 1999.

[13] J.G. Raaijmakers and R.M. Shiffrin. Search of associative memory. *Psychological Review*, 88(2):93, 1981.

[14] A. K. Romney, D. D. Brewer, and W. H. Batchelder. Predicting clustering from semantic structure. *Psychological Science*, 4(1):28–34, 1993.

[15] R. N. Shepard and P. Arabie. Additive clustering: Representation of similarities as combinations of discrete overlapping properties. *Psychological Review*, 86(2):87, 1979.

[16] D.W. Stephens and J.R. Krebs. *Foraging theory*. Princeton University Press, 1986.

[17] M. Steyvers, R.M. Shiffrin, and D.L. Nelson. Word association spaces for predicting semantic similarity effects in episodic memory. *Experimental cognitive psychology and its applications: Festschrift in honor of Lyle Bourne, Walter Kintsch, and Thomas Landauer*, pages 237–249, 2004.

[18] M. Steyvers and J.B. Tenenbaum. The large-scale structure of semantic networks: Statistical analyses and a model of semantic growth. *Cognitive Science*, 29(1):41–78, 2005.

[19] L.L. Thurstone. Primary mental abilities. *Psychometric Monographs*, 1938.

[20] A.I. Tröster, D.P. Salmon, D. McCullough, and N. Butters. A comparison of the category fluency deficits associated with Alzheimer's and Huntington's disease. *Brain and Language*, 37(3):500–513, 1989.

[21] A. K. Troyer, M. Moscovitch, and G. Winocur. Clustering and switching as two components of verbal fluency: Evidence from younger and older healthy adults. *Neuropsychology*, 11(1):138, 1997.

[22] A. K. Troyer, M. Moscovitch, G. Winocur, L. Leach, and M. Freedman. Clustering and switching on verbal fluency tests in Alzheimer's and Parkinson's disease. *Journal of the International Neuropsychological Society*, 4(2):137–143, 1998.

